# Out-of-Sample Extensions for LLE, Isomap, MDS, Eigenmaps, and Spectral Clustering

**Yoshua Bengio, Jean-François Paiement, Pascal Vincent**
**Olivier Delalleau, Nicolas Le Roux and Marie Ouimet**
Département d'Informatique et Recherche Opérationnelle
Université de Montréal
Montréal, Québec, Canada, H3C 3J7
{bengioy,vincentp,paiemeje,delallea,lerouxni,ouimema}
@iro.umontreal.ca

## Abstract

Several unsupervised learning algorithms based on an eigendecomposition provide either an embedding or a clustering only for given training points, with no straightforward extension for out-of-sample examples short of recomputing eigenvectors. This paper provides a unified framework for extending Local Linear Embedding (LLE), Isomap, Laplacian Eigenmaps, Multi-Dimensional Scaling (for dimensionality reduction) as well as for Spectral Clustering. This framework is based on seeing these algorithms as learning eigenfunctions of a data-dependent kernel. Numerical experiments show that the generalizations performed have a level of error comparable to the variability of the embedding algorithms due to the choice of training data.

## 1 Introduction

Many unsupervised learning algorithms have been recently proposed, all using an eigen-decomposition for obtaining a lower-dimensional embedding of data lying on a non-linear manifold: Local Linear Embedding (LLE) (Roweis and Saul, 2000), Isomap (Tenenbaum, de Silva and Langford, 2000) and Laplacian Eigenmaps (Belkin and Niyogi, 2003). There are also many variants of Spectral Clustering (Weiss, 1999; Ng, Jordan and Weiss, 2002), in which such an embedding is an intermediate step before obtaining a clustering of the data that can capture flat, elongated and even curved clusters. The two tasks (manifold learning and clustering) are linked because the clusters found by spectral clustering can be arbitrary curved manifolds (as long as there is enough data to locally capture their curvature).

## 2 Common Framework

In this paper we consider five types of unsupervised learning algorithms that can be cast in the same framework, based on the computation of an embedding for the training points obtained from the principal eigenvectors of a symmetric matrix.

### Algorithm 1

**1.** *Start from a data set $D = \{x_1, \ldots, x_n\}$ with $n$ points in $\mathbf{R}^d$. Construct a $n \times n$ "neighborhood" or similarity matrix $M$. Let us denote $K_D(\cdot, \cdot)$ (or $K$ for shorthand) the data-dependent function which produces $M$ by $M_{ij} = K_D(x_i, x_j)$.*

**2.** *Optionally transform $M$, yielding a "normalized" matrix $\tilde{M}$. Equivalently, this corresponds to generating $\tilde{M}$ from a $\tilde{K}_D$ by $\tilde{M}_{ij} = \tilde{K}_D(x_i, x_j)$.*

**3.** *Compute the $m$ largest positive eigenvalues $\lambda_k$ and eigenvectors $v_k$ of $\tilde{M}$.*

**4.** *The embedding of each example $x_i$ is the vector $y_i$ with $y_{ik}$ the $i$-th element of the $k$-th principal eigenvector $v_k$ of $\tilde{M}$. Alternatively (MDS and Isomap), the embedding is $e_i$, with $e_{ik} = \sqrt{\lambda_k} y_{ik}$. If the first $m$ eigenvalues are positive, then $e_i \cdot e_j$ is the best approximation of $\tilde{M}_{ij}$ using only $m$ coordinates, in the squared error sense.*

In the following, we consider the specializations of Algorithm 1 for different unsupervised learning algorithms. Let $S_i$ be the $i$-th row sum of the affinity matrix $M$:

$$S_i = \sum_j M_{ij}. \tag{1}$$

We say that two points $(a, b)$ are **k-nearest-neighbors of each other** if $a$ is among the $k$ nearest neighbors of $b$ in $D \cup \{a\}$ or vice-versa. We denote by $x_{ij}$ the $j$-th coordinate of the vector $x_i$.

### 2.1 Multi-Dimensional Scaling

Multi-Dimensional Scaling (MDS) starts from a notion of distance or affinity $K$ that is computed between each pair of training examples. We consider here metric MDS (Cox and Cox, 1994). For the normalization step 2 in Algorithm 1, these distances are converted to equivalent dot products using the "double-centering" formula:

$$\tilde{M}_{ij} = -\frac{1}{2}\left(M_{ij} - \frac{1}{n}S_i - \frac{1}{n}S_j + \frac{1}{n^2}\sum_k S_k\right). \tag{2}$$

The embedding $e_{ik}$ of example $x_i$ is given by $\sqrt{\lambda_k} v_{ki}$.

### 2.2 Spectral Clustering

Spectral clustering (Weiss, 1999) can yield impressively good results where traditional clustering looking for "round blobs" in the data, such as K-means, would fail miserably. It is based on two main steps: first embedding the data points in a space in which clusters are more "obvious" (using the eigenvectors of a Gram matrix), and then applying a classical clustering algorithm such as K-means, e.g. as in (Ng, Jordan and Weiss, 2002). The affinity matrix $M$ is formed using a kernel such as the Gaussian kernel. Several normalization steps have been proposed. Among the most successful ones, as advocated in (Weiss, 1999; Ng, Jordan and Weiss, 2002), is the following:

$$\tilde{M}_{ij} = \frac{M_{ij}}{\sqrt{S_i S_j}}. \tag{3}$$

To obtain $m$ clusters, the first $m$ principal eigenvectors of $\tilde{M}$ are computed and K-means is applied on the unit-norm coordinates, obtained from the embedding $y_{ik} = v_{ki}$.

### 2.3 Laplacian Eigenmaps

Laplacian Eigenmaps is a recently proposed dimensionality reduction procedure (Belkin and Niyogi, 2003) that has been proposed for *semi-supervised learning*. The authors use an approximation of the Laplacian operator such as the Gaussian kernel or the matrix whose element $(i, j)$ is 1 if $x_i$ and $x_j$ are k-nearest-neighbors and 0 otherwise. Instead of solving an ordinary eigenproblem, the following generalized eigenproblem is solved:

$$(S - M)v_j = \lambda_j S v_j \tag{4}$$

with eigenvalues $\lambda_j$, eigenvectors $v_j$ and $S$ the diagonal matrix with entries given by eq. (1). The smallest eigenvalue is left out and the eigenvectors corresponding to the other small eigenvalues are used for the embedding. This is the same embedding that is computed with the spectral clustering algorithm from (Shi and Malik, 1997). As noted in (Weiss, 1999) (Normalization Lemma 1), an equivalent result (up to a componentwise scaling of the embedding) can be obtained by considering the principal eigenvectors of the normalized matrix defined in eq. (3).

### 2.4 Isomap

Isomap (Tenenbaum, de Silva and Langford, 2000) generalizes MDS to non-linear manifolds. It is based on replacing the Euclidean distance by an approximation of the geodesic distance on the manifold. We define the *geodesic distance with respect to a data set D, a distance $d(u, v)$ and a neighborhood $k$* as follows:

$$\tilde{D}(a, b) = \min_p \sum_i d(p_i, p_{i+1}) \tag{5}$$

where $p$ is a sequence of points of length $l \geq 2$ with $p_1 = a$, $p_l = b$, $p_i \in D$ $\forall i \in \{2, \ldots, l-1\}$ and $(p_i, p_{i+1})$ are k-nearest-neighbors. The length $l$ is free in the minimization. The Isomap algorithm obtains the normalized matrix $\tilde{M}$ from which the embedding is derived by transforming the raw pairwise distances matrix as follows: first compute the matrix $M_{ij} = \tilde{D}^2(x_i, x_j)$ of squared geodesic distances with respect to the data $D$, then apply to this matrix the distance-to-dot-product transformation (eq. (2)), as for MDS. As in MDS, the embedding is $e_{ik} = \sqrt{\lambda_k} v_{ki}$ rather than $y_{ik} = v_{ki}$.

### 2.5 LLE

The Local Linear Embedding (LLE) algorithm (Roweis and Saul, 2000) looks for an embedding that preserves the local geometry in the neighborhood of each data point. First, a sparse matrix of local predictive weights $W_{ij}$ is computed, such that $\sum_j W_{ij} = 1$, $W_{ij} = 0$ if $x_j$ is not a k-nearest-neighbor of $x_i$ and $(\sum_j W_{ij} x_j - x_i)^2$ is minimized. Then the matrix

$$M = (I - W)'(I - W) \tag{6}$$

is formed. The embedding is obtained from the lowest eigenvectors of $M$, except for the smallest eigenvector which is uninteresting because it is $(1, 1, \ldots 1)$, with eigenvalue 0. Note that the lowest eigenvectors of $M$ are the largest eigenvectors of $\tilde{M}_\mu = \mu I - M$ to fit Algorithm 1 (the use of $\mu > 0$ will be discussed in section 4.4). The embedding is given by $y_{ik} = v_{ki}$, and is constant with respect to $\mu$.

## 3 From Eigenvectors to Eigenfunctions

To obtain an embedding for a new data point, we propose to use the Nyström formula (eq. 9) (Baker, 1977), which has been used successfully to speed-up kernel methods computations by focussing the heavier computations (the eigendecomposition) on a subset of examples. The use of this formula can be justified by considering the convergence of eigenvectors and eigenvalues, as the number of examples increases (Baker, 1977; Williams and Seeger, 2000; Koltchinskii and Giné, 2000; Shawe-Taylor and Williams, 2003). Intuitively, the extensions to obtain the embedding for a new example require specifying a new column of the Gram matrix $\tilde{M}$, through a training-set dependent kernel function $\tilde{K}_D$, in which one of the arguments may be required to be in the training set.

If we start from a data set $D$, obtain an embedding for its elements, and add more and more data, the embedding for the points in $D$ converges (for eigenvalues that are unique). (Shawe-Taylor and Williams, 2003) give bounds on the convergence error (in the case of kernel PCA). In the limit, we expect each eigenvector to converge to an eigenfunction for the linear operator defined below, in the sense that the $i$-th element of the $k$-th eigenvector converges to the application of the $k$-th eigenfunction to $x_i$ (up to a normalization factor).

Consider a Hilbert space $\mathcal{H}_p$ of functions with inner product $\langle f, g \rangle_p = \int f(x)g(x)p(x)dx$, with a density function $p(x)$. Associate with kernel $K$ a linear operator $K_p$ in $\mathcal{H}_p$:

$$(K_p f)(x) = \int K(x, y) f(y) p(y) dy. \tag{7}$$

We don't know the true density $p$ but we can approximate the above inner product and linear operator (and its eigenfunctions) using the empirical distribution $\hat{p}$. An 'empirical' Hilbert space $\mathcal{H}_{\hat{p}}$ is thus defined using $\hat{p}$ instead of $p$. Note that the proposition below can be

applied *even if the kernel is not positive semi-definite*, although the embedding algorithms we have studied are restricted to using the principal coordinates associated with positive eigenvalues. For a more rigorous mathematical analysis, see (Bengio et al., 2003).

**Proposition 1**

*Let $\tilde{K}(a, b)$ be a kernel function, not necessarily positive semi-definite, that gives rise to a symmetric matrix $\tilde{M}$ with entries $\tilde{M}_{ij} = \tilde{K}(x_i, x_j)$ upon a dataset $D = \{x_1, \ldots, x_n\}$. Let $(v_k, \lambda_k)$ be an (eigenvector,eigenvalue) pair that solves $\tilde{M} v_k = \lambda_k v_k$. Let $(f_k, \lambda'_k)$ be an (eigenfunction,eigenvalue) pair that solves $(\tilde{K}_{\hat{p}} f_k)(x) = \lambda'_k f_k(x)$ for any $x$, with $\hat{p}$ the empirical distribution over $D$. Let $e_k(x) = y_k(x)\sqrt{\lambda_k}$ or $y_k(x)$ denote the embedding associated with a new point $x$. Then*

$$\lambda'_k = \frac{1}{n}\lambda_k \tag{8}$$

$$f_k(x) = \frac{\sqrt{n}}{\lambda_k} \sum_{i=1}^{n} v_{ki} \tilde{K}(x, x_i) \tag{9}$$

$$f_k(x_i) = \sqrt{n} v_{ki} \tag{10}$$

$$y_k(x) = \frac{f_k(x)}{\sqrt{n}} = \frac{1}{\lambda_k} \sum_{i=1}^{n} v_{ki} \tilde{K}(x, x_i) \tag{11}$$

$$y_k(x_i) = y_{ik}, \qquad e_k(x_i) = e_{ik} \tag{12}$$

See (Bengio et al., 2003) for a proof and further justifications of the above formulae. The generalized embedding for Isomap and MDS is $e_k(x) = \sqrt{\lambda_k} y_k(x)$ whereas the one for spectral clustering, Laplacian eigenmaps and LLE is $y_k(x)$.

**Proposition 2**

*In addition, if the data-dependent kernel $\tilde{K}_D$ is positive semi-definite, then*

$$f_k(x) = \sqrt{\frac{n}{\lambda_k}} \pi_k(x)$$

*where $\pi_k(x)$ is the $k$-th component of the kernel PCA projection of $x$ obtained from the kernel $\tilde{K}_D$ (up to centering).*

This relation with kernel PCA (Schölkopf, Smola and Müller, 1998), already pointed out in (Williams and Seeger, 2000), is further discussed in (Bengio et al., 2003).

## 4   Extending to new Points

Using Proposition 1, one obtains a natural extension of all the unsupervised learning algorithms mapped to Algorithm 1, provided we can write down a kernel function $\tilde{K}$ that gives rise to the matrix $\tilde{M}$ on $D$, and can be used in eq. (11) to generalize the embedding. We consider each of them in turn below. In addition to the convergence properties discussed in section 3, another justification for using equation (9) is given by the following proposition:

**Proposition 3**

*If we define the $f_k(x_i)$ by eq. (10) and take a new point $x$, the value of $f_k(x)$ that minimizes*

$$\sum_{i=1}^{n} \left( \tilde{K}(x, x_i) - \sum_{t=1}^{m} \lambda'_t f_t(x) f_t(x_i) \right)^2 \tag{13}$$

*is given by eq. (9), for $m \geq 1$ and any $k \leq m$.*

The proof is a direct consequence of the orthogonality of the eigenvectors $v_k$. This proposition links equations (9) and (10). Indeed, we can obtain eq. (10) when trying to approximate

$\tilde{K}$ at the data points by minimizing the cost

$$\sum_{i,j=1}^{n} \left( \tilde{K}(x_i, x_j) - \sum_{t=1}^{m} \lambda_t' f_t(x_i) f_t(x_j) \right)^2$$

for $m = 1, 2, \ldots$ When we add a new point $x$, it is thus natural to use the same cost to approximate the $\tilde{K}(x, x_i)$, which yields (13). Note that by doing so, we do not seek to approximate $\tilde{K}(x, x)$. Future work should investigate embeddings which minimize the empirical reconstruction error of $\tilde{K}$ but ignore the diagonal contributions.

## 4.1 Extending MDS

For MDS, a normalized kernel can be defined as follows, using a continuous version of the double-centering eq. (2):

$$\tilde{K}(a, b) = -\frac{1}{2}(d^2(a, b) - E_x[d^2(x, b)] - E_{x'}[d^2(a, x')] + E_{x,x'}[d^2(x, x')]) \qquad (14)$$

where $d(a, b)$ is the original distance and the expectations are taken over the empirical data $D$. An extension of metric MDS to new points has already been proposed in (Gower, 1968), solving exactly for the embedding of $x$ to be consistent with its distances to training points, which in general requires adding a new dimension.

## 4.2 Extending Spectral Clustering and Laplacian Eigenmaps

Both the version of Spectral Clustering and Laplacian Eigenmaps described above are based on an initial kernel $K$, such as the Gaussian or nearest-neighbor kernel. An equivalent normalized kernel is:

$$\tilde{K}(a, b) = \frac{1}{n} \frac{K(a, b)}{\sqrt{E_x[K(a, x)]E_{x'}[K(b, x')]}}$$

where the expectations are taken over the empirical data $D$.

## 4.3 Extending Isomap

To extend Isomap, the test point is not used in computing the geodesic distance between training points, otherwise we would have to recompute all the geodesic distances. A reasonable solution is to use the definition of $\tilde{D}(a, b)$ in eq. (5), which only uses the training points in the intermediate points on the path from $a$ to $b$. We obtain a normalized kernel by applying the continuous double-centering of eq. (14) with $d = \tilde{D}$.

A formula has already been proposed (de Silva and Tenenbaum, 2003) to approximate Isomap using only a subset of the examples (the 'landmark" points) to compute the eigenvectors. Using our notations, this formula is

$$e_k'(x) = \frac{1}{2\sqrt{\lambda_k}} \sum_i v_{ki}(E_{x'}[\tilde{D}^2(x', x_i)] - \tilde{D}^2(x_i, x)). \qquad (15)$$

where $E_{x'}$ is an average over the data set. The formula is applied to obtain an embedding for the non-landmark examples.

**Corollary 1**

*The embedding proposed in Proposition 1 for Isomap $(e_k(x))$ is equal to formula 15 (Landmark Isomap) when $\tilde{K}(x, y)$ is defined as in eq. (14) with $d = \tilde{D}$.*

**Proof**: the proof relies on a property of the Gram matrix for Isomap: $\sum_i M_{ij} = 0$, by construction. Therefore $(1, 1, \ldots 1)$ is an eigenvector with eigenvalue 0, and all the other eigenvectors $v_k$ have the property $\sum_i v_{ki} = 0$ because of the orthogonality with $(1, 1, \ldots 1)$. Writing $(E_{x'}[\tilde{D}^2(x', x_i)] - \tilde{D}^2(x, x_i)) = 2\tilde{K}(x, x_i) + E_{x',x''}[\tilde{D}^2(x', x'')] - E_{x'}[\tilde{D}^2(x, x')]$ yields $e_k'(x) = \frac{2}{2\sqrt{\lambda_k}} \sum_i v_{ki} \tilde{K}(x, x_i) + (E_{x',x''}[\tilde{D}^2(x', x'')] - E_{x'}[\tilde{D}^2(x, x')]) \sum_i v_{ki} = e_k(x)$, since the last sum is 0.

### 4.4 Extending LLE

The extension of LLE is the most challenging one because it does not fit as well the framework of Algorithm 1: the $M$ matrix for LLE does not have a clear interpretation in terms of distance or dot product. An extension has been proposed in (Saul and Roweis, 2002), but unfortunately it cannot be cast directly into the framework of Proposition 1. Their embedding of a new point $x$ is given by

$$y_k(x) = \sum_{i=1}^{n} y_k(x_i) w(x, x_i) \tag{16}$$

where $w(x, x_i)$ is the weight of $x_i$ in the reconstruction of $x$ by its $k$-nearest-neighbors in the training set (if $x = x_j \in D$, $w(x, x_i) = \delta_{ij}$). This is very close to eq. (11), but lacks the normalization by $\lambda_k$. However, we can see this embedding as a limit case of Proposition 1, as shown below.

We first need to define a kernel $\tilde{K}_\mu$ such that

$$\tilde{K}_\mu(x_i, x_j) = \tilde{M}_{\mu,ij} = (\mu - 1)\delta_{ij} + W_{ij} + W_{ji} - \sum_k W_{ki}W_{kj} \tag{17}$$

for $x_i, x_j \in D$. Let us define a kernel $\tilde{K}'$ by

$$\tilde{K}'(x_i, x) = \tilde{K}'(x, x_i) = w(x, x_i)$$

and $\tilde{K}'(x, y) = 0$ when neither $x$ nor $y$ is in the training set $D$. Let $\tilde{K}''$ be defined by

$$\tilde{K}''(x_i, x_j) = W_{ij} + W_{ji} - \sum_k W_{ki}W_{kj}$$

and $\tilde{K}''(x, y) = 0$ when either $x$ or $y$ isn't in $D$. Then, by construction, the kernel $\tilde{K}_\mu = (\mu - 1)\tilde{K}' + \tilde{K}''$ verifies eq. (17). Thus, we can apply eq. (11) to obtain an embedding of a new point $x$, which yields

$$y_{\mu,k}(x) = \frac{1}{\lambda_k} \sum_i y_{ik} \left( (\mu - 1)\tilde{K}'(x, x_i) + \tilde{K}''(x, x_i) \right)$$

with $\lambda_k = (\mu - \hat{\lambda}_k)$, and $\hat{\lambda}_k$ being the $k$-th lowest eigenvalue of $M$. This rewrites into

$$y_{\mu,k}(x) = \frac{\mu - 1}{\mu - \hat{\lambda}_k} \sum_i y_{ik} w(x, x_i) + \frac{1}{\mu - \hat{\lambda}_k} \sum_i y_{ik} \tilde{K}''(x, x_i).$$

Then when $\mu \to \infty$, $y_{\mu,k}(x) \to y_k(x)$ defined by eq. (16).

Since the choice of $\mu$ is free, we can thus consider eq. (16) as approximating the use of the kernel $\tilde{K}_\mu$ with a large $\mu$ in Proposition 1. This is what we have done in the experiments described in the next section. Note however that we can find smoother kernels $\tilde{K}_\mu$ verifying eq. (17), giving other extensions of LLE from Proposition 1. It is out of the scope of this paper to study which kernel is best for generalization, but it seems desirable to use a smooth kernel that would take into account not only the reconstruction of $x$ by its neighbors $x_i$, but also the reconstruction of the $x_i$ by their neighbors including the new point $x$.

## 5  Experiments

We want to evaluate whether the precision of the generalizations suggested in the previous section is comparable to the intrinsic perturbations of the embedding algorithms. The perturbation analysis will be achieved by considering splits of the data in three sets, $D = F \cup R_1 \cup R_2$ and training either with $F \cup R_1$ or $F \cup R_2$, comparing the embeddings on $F$. For each algorithm described in section 2, we apply the following procedure:

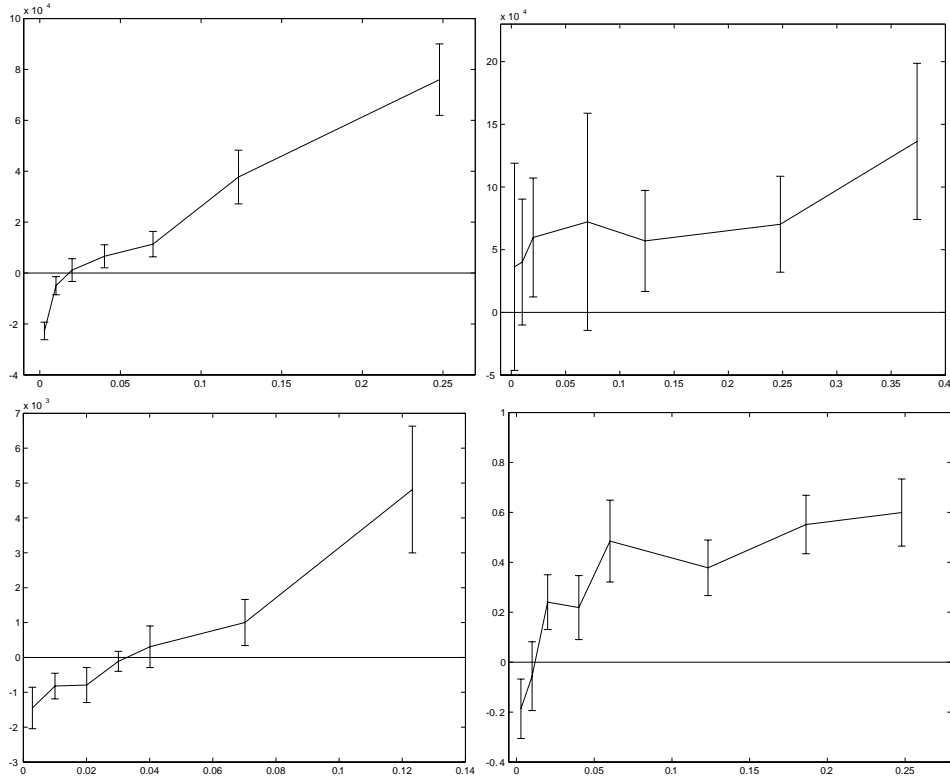

Figure 1: Training set variability minus out-of-sample error, wrt the proportion of training samples substituted. Top left: MDS. Top right: spectral clustering or Laplacian eigenmaps. Bottom left: Isomap. Bottom right: LLE. Error bars are 95% confidence intervals.

1. We choose $F \subset D$ with $m = |F|$ samples. The remaining $n - m$ samples in $D/F$ are split into two equal size subsets $R_1$ and $R_2$. We train (obtain the eigenvectors) over $F \cup R_1$ and $F \cup R_2$. When eigenvalues are close, the estimated eigenvectors are unstable and can rotate in the subspace they span. Thus we estimate an affine alignment between the two embeddings using the points in $F$, and we calculate the Euclidean distance between the aligned embeddings obtained for each $s_i \in F$.

2. For each sample $s_i \in F$, we also train over $\{F \cup R_1\}/\{s_i\}$. We apply the extension to out-of-sample points to find the predicted embedding of $s_i$ and calculate the Euclidean distance between this embedding and the one obtained when training with $F \cup R_1$, i.e. with $s_i$ in the training set.

3. We calculate the mean difference (and its standard error, shown in the figure) between the distance obtained in step 1 and the one obtained in step 2 for each sample $s_i \in F$, and we repeat this experiment for various sizes of $F$.

The results obtained for MDS, Isomap, spectral clustering and LLE are shown in figure 1 for different values of $m$. Experiments are done over a database of 698 synthetic face images described by 4096 components that is available at `http://isomap.stanford.edu`. Qualitatively similar results have been obtained over other databases such as Ionosphere (`http://www.ics.uci.edu/~mlearn/MLSummary.html`) and swissroll (`http://www.cs.toronto.edu/~roweis/lle/`). Each algorithm generates a two-dimensional embedding of the images, following the experiments reported for Isomap. The number of neighbors is 10 for Isomap and LLE, and a Gaussian kernel with a standard deviation of 0.01 is used for spectral clustering / Laplacian eigenmaps. 95% confidence

intervals are drawn beside each mean difference of error on the figure.

As expected, the mean difference between the two distances is almost monotonically increasing as the fraction of substituted examples grows (x-axis in the figure). In most cases, the out-of-sample error is less than or comparable to the training set embedding stability: it corresponds to substituting a fraction of between 1 and 4% of the training examples.

## 6  Conclusions

In this paper we have presented an extension to five unsupervised learning algorithms based on a spectral embedding of the data: MDS, spectral clustering, Laplacian eigenmaps, Isomap and LLE. This extension allows one to apply a trained model to out-of-sample points without having to recompute eigenvectors. It introduces a notion of function induction and generalization error for these algorithms. The experiments on real high-dimensional data show that the average distance between the out-of-sample and in-sample embeddings is comparable or lower than the variation in in-sample embedding due to replacing a few points in the training set.

## References

Baker, C. (1977). *The numerical treatment of integral equations*. Clarendon Press, Oxford.

Belkin, M. and Niyogi, P. (2003). Laplacian eigenmaps for dimensionality reduction and data representation. *Neural Computation*, 15(6):1373–1396.

Bengio, Y., Vincent, P., Paiement, J., Delalleau, O., Ouimet, M., and Le Roux, N. (2003). Spectral clustering and kernel pca are learning eigenfunctions. Technical report, Département d'informatique et recherche opérationnelle, Université de Montréal.

Cox, T. and Cox, M. (1994). *Multidimensional Scaling*. Chapman & Hall, London.

de Silva, V. and Tenenbaum, J. (2003). Global versus local methods in nonlinear dimensionality reduction. In Becker, S., Thrun, S., and Obermayer, K., editors, *Advances in Neural Information Processing Systems*, volume 15, pages 705–712, Cambridge, MA. The MIT Press.

Gower, J. (1968). Adding a point to vector diagrams in multivariate analysis. *Biometrika*, 55(3):582–585.

Koltchinskii, V. and Giné, E. (2000). Random matrix approximation of spectra of integral operators. *Bernoulli*, 6(1):113–167.

Ng, A. Y., Jordan, M. I., and Weiss, Y. (2002). On spectral clustering: Analysis and an algorithm. In Dietterich, T. G., Becker, S., and Ghahramani, Z., editors, *Advances in Neural Information Processing Systems 14*, Cambridge, MA. MIT Press.

Roweis, S. and Saul, L. (2000). Nonlinear dimensionality reduction by locally linear embedding. *Science*, 290(5500):2323–2326.

Saul, L. and Roweis, S. (2002). Think globally, fit locally: unsupervised learning of low dimensional manifolds. *Journal of Machine Learning Research*, 4:119–155.

Schölkopf, B., Smola, A., and Müller, K.-R. (1998). Nonlinear component analysis as a kernel eigenvalue problem. *Neural Computation*, 10:1299–1319.

Shawe-Taylor, J. and Williams, C. (2003). The stability of kernel principal components analysis and its relation to the process eigenspectrum. In Becker, S., Thrun, S., and Obermayer, K., editors, *Advances in Neural Information Processing Systems*, volume 15. The MIT Press.

Shi, J. and Malik, J. (1997). Normalized cuts and image segmentation. In *Proc. IEEE Conf. Computer Vision and Pattern Recognition*, pages 731–737.

Tenenbaum, J., de Silva, V., and Langford, J. (2000). A global geometric framework for nonlinear dimensionality reduction. *Science*, 290(5500):2319–2323.

Weiss, Y. (1999). Segmentation using eigenvectors: a unifying view. In *Proceedings IEEE International Conference on Computer Vision*, pages 975–982.

Williams, C. and Seeger, M. (2000). The effect of the input density distribution on kernel-based classifiers. In *Proceedings of the Seventeenth International Conference on Machine Learning*. Morgan Kaufmann.
